# A Novel Approach to Prediction of the 3-Dimensional Structures of Protein Backbones by Neural Networks

Henrik Fredholm[1,5]
and
Henrik Bohr[2], Jakob Bohr[3], Søren Brunak[4],
Rodney M.J. Cotterill[4], Benny Lautrup[5] and Steffen B. Petersen[1]

[1]MR-Senteret, SINTEF, N-7034 Trondheim, Norway.
[2]University of Illinois, Urbana, IL 61801, USA.
[3]Risø National Laboratory, DK-4000 Roskilde, Denmark.
[4]Technical Univ. of Denmark, B. 307, DK-2800 Lyngby, Denmark.
[5]Niels Bohr Institute, Blegdamsvej 17, DK-2100 Cph. Ø, Denmark.

## Abstract

Three-dimensional (3D) structures of protein backbones have been predicted using neural networks. A feed forward neural network was trained on a class of functionally, but not structurally, homologous proteins, using backpropagation learning. The network generated tertiary structure information in the form of binary distance constraints for the $C_\alpha$ atoms in the protein backbone. The binary distance between two $C_\alpha$ atoms was 0 if the distance between them was less than a certain threshold distance, and 1 otherwise. The distance constraints predicted by the trained neural network were utilized to generate a folded conformation of the protein backbone, using a steepest descent minimization approach.

## 1 INTRODUCTION

One current aim of molecular biology is determination of the (3D) tertiary structures of proteins in their folded native state from their sequences of amino acid

residues. Since Kendrew & Perutz solved the first protein structures, myoglobin and hemoglobin, and explained from the discovered structures how these proteins perform their function, it has been widely recognized that protein function is intimately linked with protein structure[1].

Within the last two decades X-ray crystallographers have solved the 3-dimensional (3D) structures of a steadily increasing number of proteins in the crystalline state, and recently 2D-NMR spectroscopy has emerged as an alternative method for small proteins in solution. Today approximately three hundred 3D structures have been solved by these methods, although only about half of them can be considered as truly different, and only around a hundred of them are solved at high resolution (that is, less than 2Å). The number of protein sequences known today is well over 20,000, and this number seems to be growing at least one order of magnitude faster than the number of known 3D protein structures.

Obviously, it is of great importance to develop tools that can predict structural aspects of proteins on the basis of knowledge acquired from known 3D structures.

## 1.1   THE PROTEIN FOLDING PROBLEM

It is generally accepted that most aspects of protein structure derive from the properties of the particular sequence of amino acids that make up the protein[1]. The classical experiment is that of Anfinsen *et al.* [2] who demonstrated that ribonuclease could be denatured and refolded without loss of enzymatic activity.

This has led to the formulation of the so-called protein folding problem: *given the sequence of amino acids of a protein, what will be its native folded conformation?*

## 1.2   SECONDARY STRUCTURE PREDICTION

Several methods have been developed for protein structure prediction. Most abundant are the methods for protein secondary structure prediction [3, 4, 5, 6]. These methods predict for each amino acid in the protein sequence what type of secondary structure the amino acid is part of. Several strategies have been suggested, most of which are based on statistical analysis of the occurrence of single amino acids or very short stretches of amino acids in secondary structural elements in known proteins. In general, these prediction schemes have a prediction accuracy of 50–60% for a three-category prediction of helix-, sheet- and coil conformations.

Recently neural networks have been applied to secondary structure prediction with encouraging results [7, 8, 9, 10]; on three-category prediction the accuracy is 65%; on two-catagory prediction of helix- and coil conformations the accuracy is 73%; and on a two-category prediction of turn- and coil conformations the accuracy is 71%. In all the three cases this is an improvement of the traditional methods.

## 1.3   TERTIARY STRUCTURE PREDICTION

The methods that exist for 3D structure prediction fall in three broad categories: (1) use of sequence homology with other protein with know 3D structure; (2) prediction of secondary structure units followed by the assembly of these units into a compact structure; and (3) use of empirical energy functions *ab initio* to derive the 3D structure.

No general method for 3D structure prediction exists today, and novel methods are most often documented through case stories that illustrate best or single case performance. The most successful methods so far has been those based on sequence homology; if significant sequence and functional homology exists between a protein of interest and proteins for which the 3D structures are known, it is possible (but cumbersome) to build a reasonable 3D model of the protein structure.

# 2   METHOD

We here describe a new method for predicting the 3D structure of a protein backbone from its amino acid sequence [11]. The main idea behind this approach is to use a noise tolerant representation of the protein backbone that is invariant to rotation and translation of the backbone[2], and then train a neural network to map protein sequences to this representation.

## 2.1   REPRESENTATION OF 3D BACKBONE STRUCTURES

The folded backbone structure of a protein brings residues that are distantly positioned in sequence close to each other in space. One may identify such close contacts and use them as constraints on the backbone conformation.

We define the binary distance $D(i,j)$ between two residues $i$ and $j$ as 0 if the distance between the $C_\alpha$ atom in residue $i$ and the $C_\alpha$ atom in residue $j$ is less than a given threshold and as 1 if it is above or equal to the threshold, a typical choice of threshold being 8Å. Organizing these distances as a binary distance matrix gives rise to a two dimensional representation of the protein backbone (figure 2a depict such matrix).

Most secondary motifs can be distinguished in this representation; helices appear as thickenings of the diagonal and anti-parallel and parallel sheets appear as stripes orthogonal and parallel to the diagonal.

It is possible to reconstruct the 3D backbone from the binary distance matrix representation by minimizing the "energy function",

$$E = \sum_{i \neq j} g(d_{ij}(| \tilde{C}_\alpha(i) - \tilde{C}_\alpha(j) | - \theta))$$

where $d_{ij} = 1 - 2D(i,j)$, $g(x) = 1/(1 + \exp(-x))$ and $\theta$ is the distance threshold. The initial positions of the $\tilde{C}_\alpha$ atoms are chosen at random. The motif for this

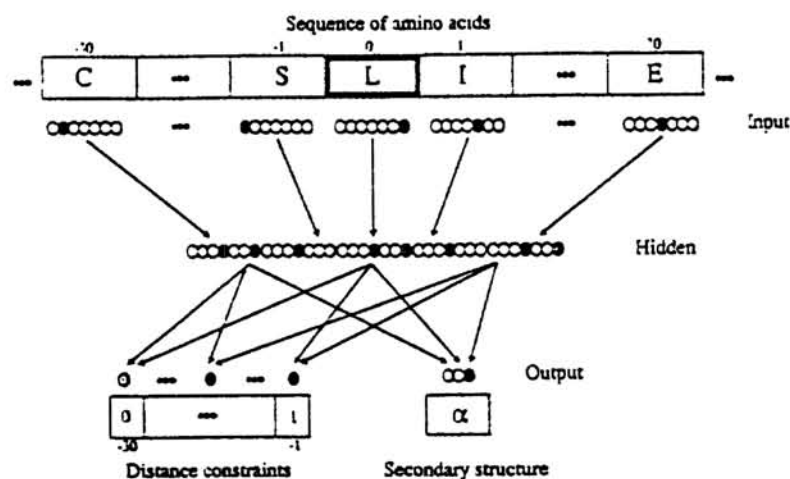

Figure 1: The input to the network consists of 61 contiguous amino acids, where each amino acid is represented by a group of 20 neurons (only seven neurons/group are illustrated). At the output layer, a set of binary distances, between the centrally positioned residue and those lying to the left of it in the input window, is produced. Secondary structure assignment for the centrally positioned residue, in the three categories of helix, sheet and coil, is also produced. Regarding the binary distance matrix, the network is trained to report which of the 30 preceding $C_\alpha$ atoms are positioned within a distance of 8Å to the centrally placed amino acid. The input layer had 1220 (61 × 20) neurons, the hidden layer had 300 neurons and the output layer had 33 neurons.

energy function is that constraints that do not hold should contribute with large values, while constraints that do hold should contribute with small values.

For small proteins of the order of 60 residues the reconstruction is very accurate. For Bovine Pancreatic Trypsin Inhibitor (6PTI), a 56 residue long protein, we were able to generate a correctly folded backbone structure. The binary distance matrix was generated from the crystallographic data of 6PTI using a distance threshold of 8Å. After convergence of the minimization procedure the errors between the reconstructed structure and the correct structure lay within 1.2Å root mean square (rms).

Preliminary results (unpublished) indicate that backbone structures for larger proteins can be reconstructed with a deviation from the correct structure down to 2Å rms, when a distance threshold of 16Å is used. When 5% random noise is added to the distance matrix the deviation from the correct structure grows to 4–5Å rms.

## 2.2 DISTANCE MATRIX PREDICTION

A backpropagation network [12] was used to map protein sequences to distance matrices. To simplify the task the neural network had to learn, it was not taught to predict all constraints in the distance matrix. Only a band along the diagonal was to be predicted. More specifically, the network was taught to predict for each residue in the protein sequence the binary distances to the 30 previous residues. Furthermore it had to classify the central residue in question as either helix, sheet or coil, see figure 1. Hence, the trained neural network produced, when given a protein sequence, a secondary structure prediction and a distance matrix containing

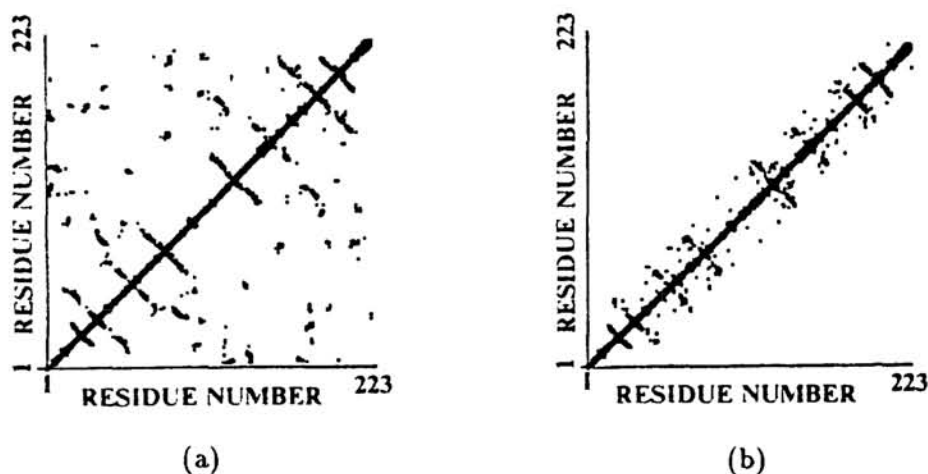

(a)                              (b)

Figure 2: Binary distance matrices for 1TRM. The matrices (223 × 223) show which $C_\alpha$ atoms are within an 8Å distance to each other $C_\alpha$ atom in the folded protein. a) The matrix corresponding to the structure determined from the X-ray data. b) Neural network prediction of an 8Å distance matrix. A 61-residue band centered along the diagonal is generated. The network predicts this band with an accuracy of 96.6%.

binary distance constraints for a lower diagonal-band matrix of width 30. Due to symmetry in the distance matrix and the diagonal being always zero, the resulting binary distance matrix contained a diagonal-band of predicted distance constraints of width 61.

## 3   CASE STORY

A neural network with this architecture was trained on 13 different proteases [13] from the Brookhaven Protein Data Bank, all having their data collected to a nominal resolution better than 2Å. The 13 proteases were of several structural classes including trypsins and subtilisins. This training set generated 3171 different examples (input windows) which were presented to the network. After 200 presentations of each example, the network had learned the training set to perfection[3]. A 14th protease, 1TRM (Rat Trypsin), with a length of 223 residues, was used to test the network. This protease was 74% homologous to one of the 13 proteases that the network was trained on. The distance matrix derived from X-ray diffraction for this protein is shown in figure 2a. The ability of the network to correctly assign structural information is amply illustrated in figure 2b, where the network is predicting the distance constraints around the diagonal for 1TRM. Although a high degree of sequence homology exists between 1TRM and the trypsins included in the training set, not a single input window presented to the network was identical to any window in the training set. The prediction thus illustrates the ability of the network to generalize from the training set. In the prediction (figure 2b), a clear distinction can be made between helices and anti-parallel sheets as well as other tertiary motifs.

If the whole binary distances matrix had been predicted, it would have been possible

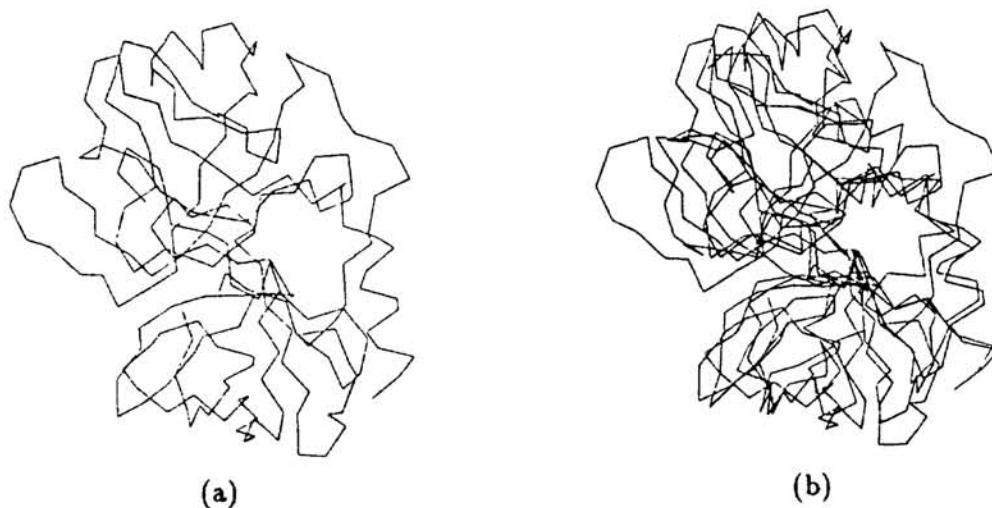

<div align="center">(a)                              (b)</div>

Figure 3: Backbone conformation for the 223 residue long trypsin 1TRM. a) The crystal structure for 1TRM, as determined by X-ray data. b) The predicted structure of 1TRM superimposed on the crystal structure. The rms deviation calculated over all the $C_\alpha$ atoms was 3Å. The largest deviations were present in surface loops, some of which are fixed by several disulphide bridges.

to construct the backbone conformation directly from the prediction. However, since only a truncated version was predicted, a good guess of the backbone conformation is needed for the minimization[4]. By using as initial guess the backbone conformation for a homologous protein, the backbone conformation of 1TRM was predicted with a 3Å rms deviation from the coordinates determined by X-ray diffraction, see figure 3. In this particular case, the length of the sequence used for the starting configuration was identical to that of the protein to be reconstructed. When the sequences are of unequal length, on the other hand, it is clear that additional considerations would have to be taken into account during the minimization process.

## 4   DISCUSSION

The single main achievement of this study has been the generation of a 3D structure of a protein from its amino acid sequence. The approach involved first the prediction of a distance matrix using a neural network and subsequently a minimization fitting procedure.

Binary distance matrices were introduced as a noise tolerant translation- and rotation invariant representation of 3D protein backbones, and a neural network was trained to map protein sequences to this representation.

The results reported here are predictions of folded conformations, illustrated with the trypsin 1TRM. Our neural network is clearly capable of generalizing the folding

information stemming from known proteins with homologous function. Current investigations have shown that the network is robust towards mutation of amino acids in the protein sequence, whereas it is very sensitive to insertions and deletions in the sequence. Thus, new network architectures will have to be developed, if this method is to be useful for proteins with low homology; a bigger training set alone will not do it.

Distance constraints can also be derived from experimental procedures such as NMR, in which they take the form of nuclear Overhauser enhancement (nOe) factors. Structural information can be successfully derived from such data using restraint dynamics which in its essential form bears some resemblance to the approach employed here, the most salient difference being that the potential energy function in our work is much simpler.

## Acknowledgements

HF thanks the Danish Research Academy, Novo-Nordisk and UNI-C for grants.

## Footnotes

[1]Although recent results indicate that certain proteins catalyze, but do not alter, the course of protein folding.

[2]The $(\phi,\psi)$ torsion-angle representation is also rotation- and translation invariant, but it is not noise tolerant.

[3]The training lasted 2 weeks on an Apollo 10000 running at 10 Mflops.

[4]For large proteins, where the band of distance constraints does not cover all spatial contacts, local folding domains may acquire different chiralities, leading to improper packing of the domains in the protein. However, new experiments indicate that the backbone structure of proteins that are 200–300 residues long can be reconstructed with good results from a random configuration, if the width of the band in the distance matrix is 121 and the distance threshold is 16Å.

## References

[1] Jaenicke, R. (1987) Prog. Biophys. Molec. Biol. **49**, 117–237.

[2] Anfinsen, C.B *et al.* (1961) Proc. Natl. Acad. Sci. USA, **47**, 1309–1314.

[3] Chou, P.Y, and Fasman, G.D. (1974) Biochemistry **13**, 211–245.

[4] Garnier, J., Osguthorpe, D.J., and Robson, B. J. (1978) Mol. Biol., **120**, 97–120.

[5] Lim, V.I. (1974) J. Mol. Biol., **88**, 857-894.

[6] Robson, D., and Suzuki, E. (1976) J. Mol. Biol., **107**, 327–356.

[7] Qian N., and Sejnowski, T.J. (1988) J. Mol. Biol., **202**, 865-884.

[8] Bohr, H., Bohr, J., Brunak, S., Cotterill, R.M.J., Lautrup, B., Nørskov, L., Olsen, O.H, and Petersen, S.B (1988) FEBS Letters, **241**, 223-228.

[9] McGregor, M.J., Flores, T.P., and Sternberg, M.J.E. (1989) Protein Engineering, **2**, 521–526.

[10] Kneller, D.G., Cohen, F.E., and Langridge, L. (1990) J. Mol. Biol., **214**, 171–182.

[11] Bohr, H., Bohr. J, Brunak, S., Cotterill, R.M.J., Fredholm, H., Lautrup, B., and Petersen, S.B. (1990) FEBS Letters, **261**, 43–46.

[12] Rummelhart, D.E., Hinton, G.E., and Williams, R.J. (1986) Parallel Distributed Processing, **1**, 318–362. Bradford Books, Cambridge, MA.

[13] Brookhaven Protein Data Bank entry codes: 1SGT (Streptomyces Trypsin), 2EST (Porcine Pancreatic Elastase), 4PTP (Bovine Pancreatic beta Trypsin), 2KAI (Porcine Pancreatic Kallikrein A), 1CHG (Bovine Chymotrypsin A), 2PRK (Fungal Proteinase K), 1SEC (Subtilisin Carlsberg), 1SGC (Streptomyces Proteinase A), 2ALP (Lysobacter Alfalytic Protease), 3APR (Rhizopus Acid Proteinase), 3RP2 (rat Mast Cell Proteinase), 2SBT (Subtilisin NOVO) and 1SAV (Subtilisin Savinase).
